# Sequential Decision Problems and Neural Networks

**A. G. Barto**
Dept. of Computer and
Information Science
Univ. of Massachusetts
Amherst, MA 01003

**R. S. Sutton**
GTE Laboratories Inc.
Waltham, MA 02254

**C. J. C. H. Watkins**
25B Framfield
Highbury, London
N5 1UU

## ABSTRACT

Decision making tasks that involve delayed consequences are very common yet difficult to address with supervised learning methods. If there is an accurate model of the underlying dynamical system, then these tasks can be formulated as sequential decision problems and solved by Dynamic Programming. This paper discusses reinforcement learning in terms of the sequential decision framework and shows how a learning algorithm similar to the one implemented by the Adaptive Critic Element used in the pole-balancer of Barto, Sutton, and Anderson (1983), and further developed by Sutton (1984), fits into this framework. Adaptive neural networks can play significant roles as modules for approximating the functions required for solving sequential decision problems.

## 1   INTRODUCTION

Most neural network research on learning assumes the existence of a supervisor or teacher knowledgeable enough to supply desired, or target, network outputs during training. These network learning algorithms are function approximation methods having various useful properties. Other neural network research addresses the question of where the training information might come from. Typical of this research is that into *reinforcement learning* systems; these systems learn without detailed

instruction about how to interact successfully with reactive environments. Learning tasks involving delays between actions and their consequences are particularly difficult to address with supervised learning methods, and special reinforcement learning algorithms have been developed to handle them. In this paper, reinforcement learning is related to the theory of sequential decision problems and to the computational methods known as Dynamic Programming (DP). DP methods are not learning methods because they rely on complete prior knowledge of the task, but their theory is nevertheless relevant for understanding and developing learning methods.

An example of a sequential decision problem invloving delayed consequences is the version of the pole-balancing problem studied by Barto, Sutton, and Anderson (1983). In this problem the consequences of control decisions are not immediately available because training information comes only in the form of a "failure signal" occurring when the pole falls past a critical angle or when the cart hits an end of the track. The learning system used by Barto et al. (1983), and subsequently systematically explored by Sutton (1984), consists of two different neuron-like adaptive elements: an Associative Search Element (ASE), which implemented and adjusted the control rule, or decision policy, and an Adaptive Critic Element (ACE), which used the failure signal to learn how to provide useful moment-to-moment evaluation of control decisions. The focus of this paper is the algorithm implemented by the ACE: What computational task does this algorithm solve, and how does it solve it?

Sutton (1988) analyzed a class of learning rules which includes the algorithm used by the ACE, calling them Temporal Difference, or TD, algorithms. Although Sutton briefly discussed the relationship between TD algorithms and DP, he did not develop this perspective. Here, we discuss an algorithm slightly different from the one implemented by the ACE and call it simply the "TD algorithm" (although the class of TD algorithms includes others as well). The earliest use of a TD algorithm that we know of was by Samuel (1959) in his checkers player. Werbos (1977) was the first we know of to suggest such algorithms in the context of DP, calling them "heuristic dynamic programming" methods. The connection to dynamic programming has recently been extensively explored by Watkins (1989), who uses the term "incremental dynamic programming." Also related is the "bucket brigade" used in classifier systems (see Liepins et al., 1989), the adaptive controller developed by Witten (1977), and certain animal learning models (see Sutton and Barto, to appear). Barto, Sutton, and Watkins (to appear) discuss the relationship between TD algorithms and DP more extensively than is possible here and provide references to other related research.

## 2    OPTIMIZING DELAYED CONSEQUENCES

Many problems require making decisions whose consequences emerge over time periods of variable and uncertain duration. Decision-making strategies must be formed that take into account expectations of both the short-term and long-term consequences of decisions. The theory of sequential decision problems is highly developed

and includes formulations of both deterministic and stochastic problems (the books by Bertsekas, 1976, and Ross, 1983, are two of the many relevant texts). This theory concerns problems such as the following special case of a stochastic problem. A decision maker (DM) interacts with a discrete-time stochastic dynamical system in such a way that, at each time step, the DM observes the system's current state and selects an action. After the action is performed, the DM receives (at the next time step) a certain amount of *payoff* that depends on the action and the current state, and the system makes a transition to a new state determined by the current state, the action, and random disturbances. Upon observing the new state, the DM chooses another action and continues in this manner for a sequence of time steps. The objective of the task is to form a rule for the DM to use in selecting actions, called a *policy*, that maximizes a measure of the total amount of payoff accumulated over time. The amount of time over which this measure is computed is the *horizon* of the problem, and a maximizing policy is an *optimal policy*. One commonly studied measure of cumulative payoff is the *expected infinite-horizon discounted return*, defined below. Because the objective is to maximize a measure of cumulative payoff, both short- and long-term consequences of decisions are important. Decisions that produce high immediate payoff may prevent high payoff from being received later on, and hence such decisions should not necessarily be included in optimal policies.

More formally (following the presentation of Ross, 1983), a policy is a mapping, denoted $\pi$, that assigns an action to each state of the underlying system (for simplicity, here we consider only the special case of deterministic policies). Let $x_t$ denote the system state at time step $t$, and if the DM uses policy $\pi$, the action it takes at step $t$ is $a_t = \pi(x_t)$. After the action is taken, the system makes a transition from state $x = x_t$ to state $y = x_{t+1}$ with a probability $P_{xy}(a_t)$. At time step $t + 1$, the DM receives a payoff, $r_{t+1}$, with expected value $R(x_t, a_t)$. For any policy $\pi$ and state $x$, one can define the expected infinite-horizon discounted return (which we simply call the *expected return*) under the condition that the system begins in state $x$, the DM continues to use policy $\pi$ throughout the future, and $\gamma$, $0 \leq \gamma < 1$, is the discount factor:

$$E_\pi \left[ \sum_{t=0}^{\infty} \gamma^t r_{t+1} | x_0 = x \right], \tag{1}$$

where $x_0$ is the initial system state, and $E_\pi$ is the expectation assuming the DM uses policy $\pi$. The objective of the decision problem is to form a policy that maximizes the expected return defined by Equation 1 for each state $x$.

## 3    DYNAMIC PROGRAMMING

Dynamic Programming (DP) is a collection of computational methods for solving stochastic sequential decision problems. These methods require a model of the dynamical system underlying the decision problem in the form of the state transition probabilities, $P_{xy}(a)$, for all states $x$ and $y$ and actions $a$, as well as knowledge of the function, $R(x, a)$, giving the payoff expectations for all states $x$ and actions $a$. There are several different DP methods, all of which are iterative methods for computing optimal policies, and all of which compute sequences of different types of *evaluation functions*. Most relevant to the TD algorithm is the evaluation function for a given

policy. This function assigns to each state the expected value of the return assuming the problem starts in that state and the given policy is used. Specifically, for policy $\pi$ and discount factor $\gamma$, the evaluation function, $V_\gamma^\pi$, assigns to each state, $x$, the expected return given the initial state $x$:

$$V_\gamma^\pi(x) = E_\pi[\sum_{t=0}^\infty \gamma^t r_{t+1}|x_0 = x].$$

For each state, the evaluation function provides a prediction of the return that will accrue throughout the future whenever this state is encountered if the given policy is followed. If one can compute the evaluation function for a state merely from observing that state, this prediction is effectively available *immediately* upon the system entering that state. Evaluation functions provide the means for assessing the temporally extended consequences of decisions in a temporally local manner.

It can be shown (e.g., Ross, 1983) that the evaluation function $V_\gamma^\pi$ is the unique function satisfying the following condition for each state $x$:

$$V_\gamma^\pi(x) = R(x, \pi(x)) + \gamma \sum_y P_{xy}(\pi(x))V_\gamma^\pi(y). \tag{2}$$

DP methods for solving this system of equations (i.e., for determining $V_\gamma^\pi$) typically proceed through successive approximations. For dynamical systems with large state sets the solution requires considerable computation. For systems with continuous state spaces, DP methods require approximations of evaluation functions (and also of policies). In their simplest form, DP methods rely on lookup-table representations of these functions, based on discretizations of the state space in continuous cases, and are therefore exponential in the state space dimension. In fact, Richard Bellman, who introduced the term Dynamic Programming (Bellman, 1957), also coined the phrase "curse of dimensionality" to describe the difficulty of representing these functions for use in DP. Consequently, any advance in function approximation methods, whether due to theoretical insights or to the development of hardware having high speed and high capacity, can be used to great advantage in DP. Artificial neural networks therefore have natural applications in DP.

Because DP methods rely on complete prior knowledge of the decision problem, they are not learning methods. However, DP methods and reinforcement learning methods are closely related, and many concepts from DP are relevant to the case of incomplete prior knowledge. Payoff values correspond to the available evaluation signals (the "primary reinforcers"), and the values of an evaluation function correspond to improved evaluation signals (the "secondary reinforcers") such a those produced by the ACE. In the simplest reinforcement learning systems, the role of the dynamical system model required by DP is played by the real system itself. A reinforcement learning system improves performance by interacting directly with the real system. A system model is not required.[1]

## 4    THE TD ALGORITHM

The TD algorithm approximates $V_\gamma^\pi$ for a given policy $\pi$ in the absence of knowledge of the transition probabilities and the function determining expected payoff values. Assume that each system state is represented by a feature vector, and that $V_\gamma^\pi$ can be approximated adequately as a function in a class of parameterized functions of the feature vectors, such as a class of functions parameterized by the connection weights of a neural network. Letting $\phi(x_t)$ denote the feature vector representing state $x_t$, let the estimated evaluation of $x_t$ be

$$V_t(x_t) = f(v_t, \phi(x_t)),$$

where $v_t$ is the weight vector at step $t$ and $f$ depends on the class of models assumed. In terms of a neural network, $\phi(x_t)$ is the input vector at time $t$, and $V_t(x_t)$ is the output at time $t$, assuming no delay across the network.

If we knew the true evaluations of the states, then we could define as an error the difference between the true evaluations and the estimated evaluations and adjust the weight vector $v_t$ according to this error using supervised-learning methods. However, it is unrealistic to assume such knowledge in sequential decision tasks. Instead the TD algorithm uses the following update rule to adjust the weight vector:

$$v_{t+1} = v_t + \alpha \left[ r_{t+1} + \gamma V_t(x_{t+1}) - V_t(x_t) \right] \frac{\partial f}{\partial v_t}(\phi(x_t)). \tag{3}$$

In this equation, $\alpha$ is a positive step-size parameter, $r_{t+1}$ is the payoff received at time step $t + 1$, $V_t(x_{t+1})$ is the estimated evaluation of the state at $t + 1$ using the weight vector $v_t$ (i.e., $V_t(x_{t+1}) = f(v_t, \phi(x_{t+1}))$),[2] and $\frac{\partial f}{\partial v_t}(\phi(x_t))$ is the gradient of $f$ with respect to $v_t$ evaluated at $\phi(x_t)$. If $f$ is the inner product of $v_t$ and $\phi(x_t)$, this gradient is just $\phi(x_t)$, as it is for a single linear ACE element. In the case of an appropriate feedforward network, this gradient can be computed by the error backpropagation method as illustrated by Anderson (1986). One can think of Equation 3 as the usual supervised-learning rule using $r_{t+1} + \gamma V_t(x_{t+1})$ as the "target" output in the error term.

To understand why the TD algorithm uses this target, assume that the DM is using a fixed policy for selecting actions. The output of the critic at time step $t$, $V_t(x_t)$, is intended to be a prediction of the return that will accrue after time step $t$. Specifically, $V_t(x_t)$ should be an estimate for the expected value of

$$r_{t+1} + \gamma r_{t+2} + \gamma^2 r_{t+3} + \cdots,$$

where $r_{t+k}$ is the payoff received at time step $t + k$. One way to adjust the weights would be to *wait forever* and use the actual return as a target. More practically,

one could wait $n$ time steps and use what Watkins (1989) calls the *n-step truncated return* as a target:

$$r_{t+1} + \gamma r_{t+2} + \gamma^2 r_{t+3} + \ldots + \gamma^{n-1} r_{t+n}.$$

However, it is possible to do better than this. One can use what Watkins calls the *corrected n-step truncated return* as a target:

$$r_{t+1} + \gamma r_{t+2} + \gamma^2 r_{t+3} + \ldots + \gamma^{n-1} r_{t+n} + \gamma^n V_t(x_{t+n}),$$

where $V_t(x_{t+n})$ is the estimated evaluation of state $x_{t+n}$ using the weight values at time $t$. Because $V_t(x_{t+n})$ is an estimate of the expected return from step $t + n + 1$ onwards, $\gamma^n V_t(x_{t+n})$ is an estimate for the missing terms in the $n$-step truncated return from state $x_t$. To see this, note that $\gamma^n V_t(x_{t+n})$ approximates

$$\gamma^n [r_{t+n+1} + \gamma r_{t+n+2} + \gamma^2 r_{t+n+3} + \ldots].$$

Multiplying through by $\gamma^n$, this equals

$$\gamma^n r_{t+n+1} + \gamma^{n+1} r_{t+n+2} + \ldots,$$

which is the part of the series missing from the $n$-step truncated return. The weight update rule for the TD algorithm (Equation 3) uses the corrected 1-step truncated return as a target, and using the $n$-step truncated return for $n > 1$ produces obvious generalizations of this learning rule at the cost of requiring longer delay lines for implementation.

The above justification of the TD algorithm is based on the assumption that the critic's output $V_t(x)$ is in fact a useful estimate of the expected return starting from any state $x$. Whether this estimate is good or bad, however, the expected value of the $n$-step corrected truncated return is always better (Watkins, 1989). Intuitively, this is true because the $n$-step corrected truncated return includes more data, namely the payoffs $r_{t+k}$, $k = 1, \ldots, n$. Surprisingly, as Sutton (1988) shows, the corrected truncated return is often a better estimate of the actual expected return than is the actual return itself.

Another way to explain the TD algorithm is to refer to the system of equations from DP (Equation 2), which the evaluation function for a given policy must satisfy. One can obtain an error based on how much the current estimated evaluation function, $V_t$, departs from the desired condition given by Equation 2 for the current state, $x_t$:

$$R(x_t, a_t) + \gamma \sum_y P_{x_t,y}(a_t) V_t(y) - V_t(x_t).$$

But the function $R$ and the transition probabilities, $P_{x_t,y}(a_t)$, are not known. Consequently, one substitutes $r_{t+1}$, the payoff actually received at step $t + 1$, for the expected value of this payoff, $R(x_t, a_t)$, and substitutes the current estimated evaluation of the state actually *reached* in one step for the expectation of the estimated evaluations of states *reachable* in one step. That is, one uses $V_t(x_{t+1})$ in place of $\sum_y P_{x_t,y}(a_t) V_t(y)$. Using the resulting error in the usual supervised-learning rule yields the TD algorithm (Equation 3).

## 5   USING THE TD ALGORITHM

We have described the TD algorithm above as a method for approximating the evaluation function associated with a fixed policy. However, if the fixed policy and the underlying dynamical system are viewed together as an autonomous dynamical system, i.e, a system without input, then the TD algorithm can be regarded purely as a prediction method, a view taken by Sutton (1988). The predicted quantity can be a discounted sum of any observable signal, not just payoff. For example, in speech recognition, the signal might give the identity of a word at the word's end, and the prediction would provide an anticipatory indication of the word's identity. Unlike other adaptive prediction methods, the TD algorithm does not require fixing a prediction time interval.

More relevant to the topic of this paper, the TD algorithm can be used as a component in methods for improving policies. The pole-balancing system of Barto et al. (1983; see also Sutton, 1984) provides one example in which the policy changes while the TD algorithm operates. The ASE of that system changes the policy by attempting to improve it according to the current estimated evaluation function. This approach is most closely related to the policy improvement algorithm of DP (e.g., see Bertsekas, 1976; Ross, 1983) and is one of several ways to use TD-like methods for improving policies; others are described by Watkins (1989) and Werbos (1987).

## 6   CONCLUSION

Decision making problems involving delayed consequences can be formulated as stochastic sequential decision problems and solved by DP if there is a complete and accurate model of the underlying dynamical system. Due to the computational cost of exact DP methods and their reliance on complete and exact models, there is a need for methods that can provide approximate solutions and that do not require this amount of prior knowledge. The TD algorithm is an incremental, on-line method for approximating the evaluation function associated with a given policy that does not require a system model. The TD algorithm directly adjusts a parameterized model of the evaluation function—a model that can take the form of an artificial neural network. The TD learning process is a Monte-Carlo approximation to a successive approximation method of DP. This perspective provides the necessary framework for extending the theory of TD algorithms as well as that of other algorithms used in reinforcement learning. Adaptive neural networks can play significant roles as modules for approximating the required functions.

### Acknowledgements

A. G. Barto's contribution was supported by the Air Force Office of Scientific Research, Bolling AFB, through grants AFOSR-87-0030 and AFOSR-89-0526.

## Footnotes

[1] Although reinforcement learning methods can greatly benefit from such models (Sutton, to appear).

[2] Instead of using $v_t$ to evaluate the state at $t+1$, the learning rule used by the ACE by Barto et al. (1983) uses $v_{t+1}$. This closely approximates the algorithm described here if the weights change slowly.

### References

C. W. Anderson. (1986) *Learning and Problem Solving with Multilayer Connectionist Systems*. PhD thesis, University of Massachusetts, Amherst, MA.

A. G. Barto, R. S. Sutton, and C. W. Anderson. (1983) Neuronlike elements that can solve difficult learning control problems. *IEEE Transactions on Systems, Man, and Cybernetics*, 13:835–846.

A. G. Barto, R. S. Sutton, and C. Watkins. (to appear) Learning and sequential decision making. In M. Gabriel and J. W. Moore, editors, *Learning and Computational Neuroscience*. The MIT Press, Cambridge, MA.

R. E. Bellman. (1957) *Dynamic Programming*. Princeton University Press, Princeton, NJ.

D. I. Bertsekas. (1976) *Dynamic Programming and Stochastic Control*. Academic Press, New York.

Liepins, G. E., Hilliard, M.R., Palmer, M., and Rangarajan, G. (1989) Alternatives for classifier system credit assignment. *Proceedings of the Eleventh International Joint Conference on Artificial Intelligence*, 756–761.

S. Ross. (1983) *Introduction to Stochastic Dynamic Programming*. Academic Press, New York.

A. L. Samuel. (1959) Some studies in machine learning using the game of checkers. *IBM Journal on Research and Development*, 210–229.

R. S. Sutton. (1984) *Temporal Credit Assignment in Reinforcement Learning*. PhD thesis, University of Massachusetts, Amherst, MA.

R. S. Sutton. (1988) Learning to predict by the methods of temporal differences. *Machine Learning*, 3:9–44.

R. S. Sutton (to appear) First results with Dyna, an integrated architecture for learning planning and reacting. *Proceedings of the 1990 AAAI Symposium on Planning in Uncertain, Unpredictable, or Changing Environments*.

R. S. Sutton and A. G. Barto. (to appear) Time-derivative models of Pavlovian reinforcement. In M. Gabriel and J. W. Moore, editors, *Learning and Computational Neuroscience*. The MIT Press, Cambridge, MA.

C. J. C. H. Watkins. (1989) *Learning from Delayed Rewards*. PhD thesis, Cambridge University, Cambridge, England.

P. J. Werbos. (1977) Advanced forecasting methods for global crisis warning and models of intelligence. *General Systems Yearbook*, 22:25–38.

P. J. Werbos. (1987) Building and understanding adaptive systems: A statistical/numerical approach to factory automation and brain research. *IEEE Transactions on Systems, Man, and Cybernetics*, 17:7–20.

I. H. Witten. (1977). An adaptive optimal controller for discrete-time markov environments. *Information and Control*, 34:286–295.
